# Automatic Derivation of Statistical Algorithms: The EM Family and Beyond

**Alexander G. Gray**
Carnegie Mellon University
agray@cs.cmu.edu

**Bernd Fischer and Johann Schumann**
RIACS / NASA Ames
{fisch,schumann}@email.arc.nasa.gov

**Wray Buntine**
Helsinki Institute for IT
buntine@hiit.fi

## Abstract

Machine learning has reached a point where many probabilistic methods can be understood as variations, extensions and combinations of a much smaller set of abstract themes, e.g., as different instances of the EM algorithm. This enables the systematic derivation of algorithms customized for different models. Here, we describe the AUTOBAYES system which takes a high-level statistical model specification, uses powerful symbolic techniques based on *schema-based program synthesis* and *computer algebra* to derive an efficient specialized algorithm for learning that model, and generates executable code implementing that algorithm. This capability is far beyond that of code collections such as Matlab toolboxes or even tools for model-independent optimization such as BUGS for Gibbs sampling: complex new algorithms can be generated without new programming, algorithms can be highly specialized and tightly crafted for the exact structure of the model and data, and efficient and commented code can be generated for different languages or systems. We present automatically-derived algorithms ranging from closed-form solutions of Bayesian textbook problems to recently-proposed EM algorithms for clustering, regression, and a multinomial form of PCA.

## 1  Automatic Derivation of Statistical Algorithms

**Overview.** We describe a symbolic program synthesis system which works as a "statistical algorithm compiler:" it compiles a statistical model specification into a custom *algorithm design* and from that further down into a working program implementing the algorithm design. This system, AUTOBAYES, can be loosely thought of as "part theorem prover, part Mathematica, part learning textbook, and part Numerical Recipes." It provides much more flexibility than a fixed code repository such as a Matlab toolbox, and allows the creation of efficient algorithms which have never before been implemented, or even written down. AUTOBAYES is intended to automate the more routine application of complex methods in novel contexts. For example, recent multinomial extensions to PCA [2, 4] can be derived in this way.

**The algorithm design problem.** Given a dataset and a task, creating a learning method can be characterized by two main questions: 1. What is the model? 2. What algorithm will optimize the model parameters? The *statistical algorithm* (i.e., a parameter optimization algorithm for the statistical model) can then be implemented manually. The system in this paper answers the algorithm question given that the user has chosen a model for the data,and continues through to implementation. Performing this task at the state-of-the-art level requires an intertwined meld of probability theory, computational mathematics, and software engineering. However, a number of factors unite to allow us to solve the algorithm design problem computationally: 1. The existence of fundamental building blocks (e.g., standardized probability distributions, standard optimization procedures, and generic data structures). 2. The existence of common representations (i.e., graphical models [3, 13] and program *schemas*). 3. The formalization of schema applicability constraints as *guards*.[1]

**The challenges of algorithm design.** The design problem has an inherently combinatorial nature, since subparts of a function may be optimized recursively and in different ways. It also involves the use of new data structures or approximations to gain performance. As the research in statistical algorithms advances, its creative focus should move beyond the ultimately mechanical aspects and towards extending the abstract applicability of already existing schemas (algorithmic principles like EM), improving schemas in ways that generalize across anything they can be applied to, and inventing radically new schemas.

## 2    Combining Schema-based Synthesis and Bayesian Networks

```
1 model mog as 'Mixture of Gaussians';

2 const int n_points as 'nr. of data points'
3   with 0 < n_points;
4 const int n_classes := 3 as 'nr. classes'
5   with 0 < n_classes
6   with n_classes << n_points;

7 double phi(1..n_classes) as 'weights'
8   with 1 = sum(I := 1..n_classes, phi(I));
9 double mu(1..n_classes);
9 double sigma(1..n_classes);

10 int c(1..n_points) as 'class labels';
11 c ~ disc(vec(I := 1..n_classes, phi(I)));

12 data double x(1..n_points) as 'data';
13 x(I) ~ gauss(mu(c(I)), sigma(c(I)));

14 max pr(x|{phi,mu,sigma}) wrt {phi,mu,sigma};
```

**Statistical Models.** Externally, AUTOBAYES has the look and feel of a compiler. Users specify their model of interest in a high-level specification language (as opposed to a *programming* language). The figure shows the specification of the mixture of Gaussians example used throughout this paper.[2] Note the constraint that the sum of the class probabilities must equal one (line 8) along with others (lines 3 and 5) that make optimization of the model well-defined. Also note the ability to specify assumptions of the kind in line 6, which may be used by some algorithms. The last line specifies the goal inference task: maximize the conditional probability $\mathrm{pr}(\vec{x}|\{\vec{\phi}, \vec{\mu}, \vec{\sigma}\})$ with respect to the parameters $\vec{\phi}$, $\vec{\mu}$, and $\vec{\sigma}$. Note that moving the parameters across to the left of the conditioning bar converts this from a maximum likelihood to a *maximum a posteriori* problem.

**Computational logic and theorem proving.** Internally, AUTOBAYES uses a class of techniques known as computational logic which has its roots in automated theorem proving. AUTOBAYES begins with an initial goal and a set of initial assertions, or axioms, and adds new assertions, or theorems, by repeated application of the axioms, until the goal is proven. In our context, the goal is given by the input model; the derived algorithms are side effects of constructive theorems proving the existence of algorithms for the goal.

**Computer algebra.** The first core element which makes automatic algorithm derivation feasible is the fact that we can mechanize the required symbol manipulation, using computer algebra methods. General *symbolic differentiation* and *expression simplification* are capabilities fundamental to our approach. AUTOBAYES contains a computer algebra engine using term rewrite rules which are an efficient mechanism for substitution of equal quantities or expressions and thus well-suited for this task.[3]

**Schema-based synthesis.** The computational cost of full-blown theorem proving grinds simple tasks to a halt while elementary and intermediate facts are reinvented from scratch. To achieve the scale of deduction required by algorithm derivation, we thus follow a *schema-based synthesis* technique which breaks away from strict theorem proving. Instead, we formalize high-level domain knowledge, such as the general EM strategy, as *schemas*. A schema combines a generic code fragment with explicitly specified preconditions which describe the applicability of the code fragment. The second core element which makes automatic algorithm derivation feasible is the fact that we can use Bayesian networks to efficiently encode the preconditions of complex algorithms such as EM.

**First-order logic representation of Bayesian networks.** A first-order logic representation of Bayesian networks was developed by Haddawy [7]. In this framework, random variables are represented by functor symbols and indexes (i.e., specific instances of i.i.d. vectors) are represented as functor arguments. Since unknown index values can be represented by implicitly universally quantified Prolog variables, this 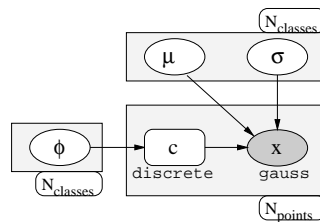
approach allows a compact encoding of networks involving i.i.d. variables or *plates* [3]; the figure shows the initial network for our running example. Moreover, such networks correspond to backtrack-free datalog programs, allowing the dependencies to be efficiently computed. We have extended the framework to work with non-ground probability queries since we seek to determine probabilities over entire i.i.d. vectors and matrices. Tests for independence on these indexed Bayesian networks are easily developed in Lauritzen's framework which uses ancestral sets and set separation [9] and is more amenable to a theorem prover than the double negatives of the more widely known d-separation criteria. Given a Bayesian network, some probabilities can easily be extracted by enumerating the component probabilities at each node:

**Lemma 1.** *Let $U, V$ be sets of variables over a Bayesian network with $U \cap V = \emptyset$. Then $V \cap descendents(U) = \emptyset$ and $parents(U) \subseteq V$ hold[4]in the corresponding dependency graph iff the following probability statement holds:*

$$Pr(U|V) = Pr(U|parents(U)) = \prod_{u \in U} Pr(u|parents(u)).$$

**Symbolic probabilistic inference.** How can probabilities not satisfying these conditions be converted to symbolic expressions? While many general schemes for inference on networks exist, our principal hurdle is the need to perform this over symbolic expressions incorporating real and integer variables from disparate real or infinite-discrete distributions. For instance, we might wish to compute the full *maximum a posteriori* probability for the mean and variance vectors of a Gaussian mixture model under a Bayesian framework. While the sum-product framework of [8] is perhaps closer to our formulation, we have out of necessity developed another scheme that lets us extract probabilities on a large class of mixed discrete and real, potentially indexed variables, where no integrals are needed and

all marginalization is done by summing out discrete variables. We give the non-indexed case below; this is readily extended to indexed variables (i.e., vectors).

**Lemma 2.** *$V \cap descendents(U) = \emptyset$ holds and $ancestors(V)$ is independent of $U$ given $V$ iff there exists a set of variables $U'$ such that Lemma 1 holds if we replace $U$ by $U \cup U'$. Moreover, the unique minimal set $U'$ satisfying these conditions is given by $ancestors(U) / (ancestors(V) \cup V)$.*

**Lemma 3.** *Let $V'$ be a subset of $V/descendents(U)$ such that $ancestors(V')$ is independent of $(U \cup V)/(V' \cup ancestors(V'))$ given $V'$. Then Lemma 2 holds if we replace $U$ by $U \cup V/V'$ and $V$ by $V'$. Moreover, there is a unique maximal set $V'$ satisfying these conditions.*

Lemma 2 lets us evaluate a probability by a summation:

$$Pr(U \mid V) = \sum_{u' \in \mathrm{Dom}(U')} Pr(U' = u', U \mid V)$$

while Lemma 3 lets us evaluate a probability by a summation and a ratio:

$$Pr(U \mid V) = \frac{Pr(U \cup V/V' \mid V')}{Pr(V/V' \mid V')}$$

Since the lemmas also show minimality of the sets $U'$ and $V/V'$, they also give the minimal conditions under which a probability can be evaluated by discrete summation without integration. These inference lemmas are operationalized as network decomposition schemas. However, we usually attempt to decompose a probability into independent components before applying this schema.

## 3   The AutoBayes System — Implementation Outline

**Levels of representation.** Internally, our system uses three conceptually different levels of representation. *Probabilities* (including logarithmic and conditional probabilities) are the most abstract level. They are processed via methods for Bayesian network decomposition or match with core algorithms such as EM. *Formulae* are introduced when probabilities of the form $Pr(U \mid parents(U))$ are detected, either in the initial network, or after the application of network decompositions. Atomic probabilities (i.e., $U$ is a single variable) are directly replaced by formulae based on the given distribution and its parameters. General probabilities are decomposed into sums and products of the respective atomic probabilities. Formulae are ready for immediate optimization using symbolic or numeric methods but sometimes they can be decomposed further into independent subproblems. Finally, we use imperative *intermediate code* as the lowest level to represent both program fragments within the schemas as well as the completely constructed programs. All transformations we apply operate on or between these levels.

**Transformations for optimization.** A number of different kinds of transformations are available. *Decomposition* of a problem into independent subproblems is always done. Decomposition of probabilities is driven by the Bayesian network; we have a separate system for handling decomposition of formulae. A formula can be decomposed along a loop, e.g., the problem "optimize $\vec{\theta}$ for $\prod_i f(\theta_i)$" is transformed into a for-loop over subproblems "optimize $\theta_i$ for $f(\theta_i)$." More commonly, "optimize $\theta, \phi$ for $f(\theta) + g(\phi)$" is transformed into the two subprograms "optimize $\theta$ for $f(\theta)$" and "optimize $\phi$ for $g(\phi)$." The lemmas given earlier are applied to change the level of representation and are thus used for *simplification of probabilities*. Examples of general *expression simplification* include simplifying the log of a formula, moving a summation inwards, and so on. When necessary, *symbolic differentiation* is performed. In the initial specification or in intermediate representations,

*likelihoods* (i.e., subexpressions of the form $log \prod_i Pr(x_i \mid \theta)$) are identified and sim­plified into linear expression with terms such as $mean(x_i)$ and $mean(x_i^2)$. The *statistical algorithm schemas* currently implemented include EM, k-means, and discrete model se­lection. Adding a Gibbs sampling schema would yield functionality comparable to that of BUGS [14]. Usually, the schemas require a particular form of the probabilities involved; they are thus tightly coupled to the decomposition and simplification transformations. For example, EM is a way of dealing with situation where Lemma 2 applies but where $U'$ is indexed identically to the data.

**Code and test generation.** From the intermediate code, code in a particular *target lan­guage* may be generated. Currently, AUTOBAYES can generate C++ and C which can be used in a stand-alone fashion or linked into Octave or Matlab (as a mex file). During this code-generation phase, most of the vector and matrix expressions are converted into for-loops, and various *code optimizations* are performed which are impossible for a standard compiler. Our tool does not only generate *efficient* code, but also highly readable, *doc­umented* programs: model- and algorithm-specific comments are generated automatically during the synthesis phase. For most examples, roughly 30% of the produced lines are comments. These comments provide explanation of the algorithm's derivation. A gener­ated HTML software design document with navigation capabilities facilitates code under­standing and reading. AUTOBAYES also automatically generates a program for *sampling* from the specified model, so that closed-loop testing with synthetic data of the assumed distributions can be done. This can be done using simple forward sampling.

## 4 Example: Deriving the EM Algorithm for Gaussian Mixtures

**1. User specifies model.** First, the user specifies the model as shown in Section 2.

**2. System parses model to obtain underlying Bayes net.** From the model, the underlying Bayesian network is derived and represented internally as a directed graph. For visualiza­tion, AUTOBAYES can also produce a graph drawing as shown in Section 2.

**3. System observes hidden-variable structure in Bayesian network.** The system at­tempts to decompose the optimization goal into independent parts, but finds that it cannot. However, it then finds that the probability in the initial optimization statement matches the conditions of Lemma 2 and that the network describes a latent variable model.

**4. System invokes abstract EM-family schema.** This triggers the EM-schema, whose overall structure is shown. The syntactic structure of the current subproblem must match the first argument of the schema; if additional applicability constraints (not shown here) hold, this schema is

$$\text{schema}(\texttt{max } Pr(U \mid V)\texttt{wrt } V, C) \; : -$$
$$\cdots$$
$$C = \text{'}[\text{initialize } W];$$
$$\quad \texttt{while}([\texttt{converging}(V)])\{$$
$$\quad \quad \text{/* M-step */ } [\text{max } Pr(W, U \mid V) \texttt{ wrt } V];$$
$$\quad \quad \text{/* E-step */ } [\text{calculate } Pr(W \mid U, V)];$$
$$\quad \}\text{''}$$

executed. It constructs a piece of code which is returned in the variable $C$. This code frag­ment can contain recursive calls to other schemas (denoted by $[\ldots]$) which return code for subproblems which then is inserted into the schema, such as $\texttt{converging}$, a generic con­vergence criterion here imposed over the variables $\vec{\mu}, \vec{\sigma}, \vec{\phi}$. Note that the schema actually implements an ME-algorithm (i.e., starts the loop with the M-step) because the initial­ization already serves as an E-step. The system identifies the discrete variable $\vec{c}$ as the single hidden variable, i.e., $W = \{\vec{c}\}$. For representation of the distribution of the hidden variable a matrix $\vec{q}$ is generated, where $q_{ij}$ is the probability that the $i$-th point falls into the $j$-th class. AUTOBAYES then constructs the new distribution $\texttt{c(I)} \; \tilde{} \; \texttt{disc(vec(J := 1..n\_classes, q(I, J))}$ which replaces the original distribution in the following recursive calls of AUTOBAYES.

```
while(converging(μ⃗, σ⃗, φ⃗)){
  for i = 1 : N
    for j = 1 : C
      q_{ij} = Pr(c_i = j|x_i, μ⃗, σ⃗);
    max Pr(x_i, c_i|μ_{c_i}, σ_{c_i}, φ⃗) wrt {μ⃗, σ⃗, φ⃗}
}
```

**5. E-step: System performs marginalization.** The freshly introduced distribution for $c_i$ implies that $c_i$ can be eliminated from the objective function by summing over $q_{i,*}$. This gives us the partial program shown in the internal pseudocode.

```
while(converging(μ⃗, σ⃗, φ⃗)){
  for i = 1 : N
    for j = 1 : C
      q_{ij} = Pr(c_i = j|x_i, μ⃗, σ⃗);
    for j = 1 : C
      max ∑_{i=1}^{N} q_{ij} log Pr(x_i|μ_j, σ_j) wrt {μ_j, σ_j}
      max ∑_{j=1}^{C}(∑_{i=1}^{N} q_{ij})φ_j wrt {φ⃗}
}
```

**6. M-step: System recursively decomposes optimization problem.** AUTOBAYES is recursively called with the new goal $\max \log Pr(\{\vec{c}, \vec{x}\} \mid \{\vec{\phi}, \vec{\mu}, \vec{\sigma}\})$ wrt $\{\vec{\phi}, \vec{\mu}, \vec{\sigma}\}$. Now, the Bayesian network decomposition schema applies with $U = \{\vec{c}, \vec{x}\}$, $V = \{\vec{\phi}, \vec{\mu}, \vec{\sigma}\}$, revealing that $\vec{\phi}$ is independent of $\vec{\sigma}, \vec{\mu}$, thus the optimization problem can be decomposed into two optimization subproblems: $\max Pr(\vec{x} \mid \{\vec{c}, \vec{\mu}, \vec{\sigma}\})$ wrt $\{\vec{\mu}, \vec{\sigma}\}$ and $\max Pr(\vec{c}|\vec{\phi})$ wrt $\{\vec{\phi}\}$.

**7. System unrolls i.i.d. vectors.** The first subgoal from the decomposition schema, $\max Pr(\vec{x} \mid \{\vec{c}, \vec{\mu}, \vec{\sigma}\})$ wrt $\{\vec{\mu}, \vec{\sigma}\}$, can be unrolled over the independent and identically distributed vector $\vec{x}$ using an index decomposition schema which moves expressions out of loops (sums or products) when they are not dependent on the loop index. Since $\vec{c}$ and $\vec{x}$ are co-indexed, unrolling proceeds over both (also independent and identically distributed) vectors in parallel: $\max \prod_{i=1}^{N} Pr(x_i \mid \{c_i, \vec{\mu}, \vec{\sigma}\})$ wrt $\{\vec{\mu}, \vec{\sigma}\}$.

**8. System identifies and solves Gaussian elimination problem.** The probability $Pr(x_i \mid \{c_i, \vec{\mu}, \vec{\sigma}\})$ is atomic because $parents(x_i) = \{c_i, \vec{\mu}, \vec{\sigma}\}$. It can thus be replaced by the appropriately instantiated Gaussian density function. Because the strictly monotone $\log(\cdot)$ function can first be applied to the objective function of the maximization, it becomes $\max \sum_{i=1}^{N} \sum_{j=1}^{C} q_{ij}(-\frac{1}{2\sigma_j}(x_i - \mu_j)^2 - \log\sqrt{2\pi} - \log\sigma_j)$ wrt $\{\vec{\mu}, \vec{\sigma}\}$. Another application of index decomposition allows solution for the two scalars $\mu_j$ and $\sigma_j$. Gaussian elimination is then used to solve this subproblem analytically, yielding the sequence of expressions $\mu_j = \sum_{i=1}^{N} q_{ij}x_i / \sum_{i=1}^{N} q_{ij}$ and $\sigma_j = \sum_{i=1}^{N} q_{ij}(x_i - \mu_j)^2 / \sum_{i=1}^{N} q_{ij}$.

**9. System identifies and solves Lagrange multiplier problem.** The second subgoal $\max Pr(\vec{c} \mid \vec{\phi})$ wrt $\{\vec{\phi}\}$ can be unrolled over the i.i.d. vector $\vec{c}$ as before. The specification condition $\sum_{j=1}^{C} \phi_j = 1$ creates a constrained maximization problem in the vector $\vec{\phi}$ which is solved by an application of the Lagrange multiplier schema. This in turn results in two subproblems for a single instance $\phi_j$ and for the multiplier which are both solved symbolically. Thus, the usual EM algorithm for Gaussian mixtures is derived.

**10. System checks and optimizes pseudocode.** During the synthesis process, AUTOBAYES accumulates a number of constraints which have to hold to ensure proper operation of the code (e.g., absence of divide-by-zero errors). Unless these constraints can be resolved against the model (e.g., $\sigma_i > 0$), AUTOBAYES automatically inserts run-time checks into the code. Before finally generating C/C++ code, the pseudocode is optimized using information from the specification (e.g., $\sum_{j=1}^{C} \phi_j = 1$) and the domain. Thus, optimizations well beyond the capability of a regular compiler can be done.

**11. System translates pseudocode to real code in desired language.** Finally, AUTOBAYES converts the intermediate code into code of the desired target system. The source code contains thorough comments detailing the mathematics implemented. A regular compiler containing generic performance optimizations not repeated by AUTOBAYES turns the code into an executable program. A program for sampling from a mixture of Gaussians is also produced for testing purposes.

# 5   Range of Capabilities

Here, we discuss 18 examples which have been successfully handled by AUTOBAYES, ranging from simple textbook examples to sophisticated EM models and recent multinomial versions of PCA. For each entry, the table below gives a brief description, the number of lines of the specification and synthesized C++ code (loc), and the runtime to generate the code (in secs., measured on a 2.2GHz Linux system). Correctness was checked for these examples using automatically-generated test data and hand-written implementations.

**Bayesian textbook examples.** Simple textbook examples, like Gaussian with simple prior $B_1$, Gaussian with inverse gamma prior $B_2$, or Gaussian with conjugate prior $B_3$ have closed-form solutions. The symbolic system of AUTOBAYES can actually find these solutions and thus generate short and efficient code. However, a slight relaxation of the prior on $\mu$ (Gaussian with semi-conjugate prior, $B_4$) requires an iterative numerical solver.

**Gaussians in action.** $G_1$ is a Gaussian change-detection model. A slight extension of our running example, integrating several features, yields a Gaussian Bayes classifier model $G_2$. $G_2$ has been successfully tested on various standard benchmarks [1], e.g., the Abalone dataset. Currently, the number of expected classes has to be given in advance.

**Mixture models and EM.** A wide range of $k$-Gaussian mixture models can be handled by AUTOBAYES, ranging from the simple 1D ($M_1$) and 2D with diagonal covariance ($M_2$) to 1D models for multi-dimensional classes $M_3$ and with (conjugate) priors on mean $M_4$ or variance $M_5$. Using only a slight variation in the specification, the Gaussian distribution can be replaced by other distributions (e.g., exponentials, $M_6$, for failure analysis) or combinations (e.g., . Gaussian and Beta, $M_7$, or $k$-Cauchy and Poisson $M_8$). In the algorithm generated by $M_7$, the analytic subsolution for the Gaussian case is combined with the numerical solver. Finally, $M_9$ is a $k_1$-Gaussians and $k_2$-Gaussians two-level hierarchical mixture model which is solved by a nested instantiation of EM [15]: i.e., the M-step of the outer EM algorithm is a second EM algorithm nested inside.

**Mixtures for Regression.** We represented regression with Gaussian error and Legendre polynomials with full conjugate priors allowing smoothing [10]. Two versions of this were then done: robust linear regression $R_1$ replaces the Gaussian error with a mixture of two Gaussians (one broad, one peaked) both centered at zero. Trajectory clustering $R_2$ replaces the single regression curve by a mixture of several curves [5]. In both cases an EM algorithm is correctly integrated with the exact regression solutions.

**Principal Component Analysis.** We also tested a multinomial version of PCA called latent Dirichlet allocation [2]. AUTOBAYES currently lacks variational support, yet it manages to combine a $k$-means style outer loop on the component proportions with an EM-style inner loop on the hidden counts, producing the original algorithm of Hofmann, Lee and Seung, and others [4].

| # | Description | loc | $T_s$ | # | Description | loc | $T_s$ |
|---|---|---|---|---|---|---|---|
| $B_1$ | $\mu \sim \mathrm{N}(\mu_0, \tau_0^{0.5})$ $\sigma^2$ | 12/137 | 0.2 | $B_2$ | $\mu$ $\sigma^2 \sim \Gamma^{-1}(\frac{\delta_0}{2}+1, \frac{\delta_0}{2}\sigma_0^{.5})$ | 13/148 | 0.2 |
| $B_3$ | $\mu \sim \mathrm{N}(\mu_0, (\frac{\sigma^2}{\kappa_0})^{0.5})$ $\sigma^2 \sim \Gamma^{-1}(\frac{\delta_0}{2}+1, \frac{\delta_0}{2}\sigma_0^{.5})$ | 16/188 | 0.4 | $B_4$ | $\mu \sim \mathrm{N}(\mu_0, \tau_0)$ $\sigma^2 \sim \Gamma^{-1}(\frac{\delta_0}{2}+1, \frac{\delta_0}{2}\sigma_0^{.5})$ | 17/233 | 0.4 |
| $G_1$ | Gauss step-detect | 19/662 | 2.0 | $G_2$ | Gauss Bayes Classify | 58/1598 | 4.7 |
| $M_1$ | $k$-Gauss mix 1D | 17/418 | 0.7 | $M_2$ | $k$-Gauss mix 2D, diag | 22/599 | 1.2 |
| $M_3$ | –"–, multi-dim | 24/900 | 1.1 | $M_4$ | –"– 1D, $\mu$ prior | 25/456 | 1.0 |
| $M_5$ | –"–, $\sigma$ prior | 21/442 | 0.9 | $M_6$ | $k$-Exp mix | 15/347 | 0.5 |
| $M_7$ | Gauss/Beta mix | 22/834 | 1.7 | $M_8$ | $k$-Cauchy/Poisson | 21/747 | 1.0 |
| $M_9$ | $k_1, k_2$-Gauss hierarch | 29/1053 | 2.3 | | mix | | |
| $R_1$ | rob. lin. regression | 54/1877 | 14.5 | $P_1$ | PCA mult/w $k$-means | 26/390 | 1.2 |
| $R_2$ | mixture regression | 53/1282 | 9.8 | | | | |

# 6   Conclusion

**Beyond existing systems.** Code libraries are common in statistics and learning, but they lack the high level of automation achievable only by deep symbolic reasoning. The Bayes Net Toolbox [12] is a Matlab library which allows users to program in models but does not derive algorithms or generate code. The BUGS system [14] also allows users to program in models but is specialized for Gibbs sampling. Stochastic parametrized grammars [11] allow a concise model specification similar to AUTOBAYES's specification language, but are currently only a notational device similar to XML.

**Benefits of automated algorithm and code generation.** *Industrial-strength code.* Code generated by AUTOBAYES is efficient, validated, and commented. *Extreme applications.* Extremely complex or critical applications such as spacecraft challenge the reliability limits of human-developed software. Automatically generated software allows for pervasive condition checking and correctness-by-construction. *Fast prototyping and experimentation.* For both the data analyst and machine learning researcher, AUTOBAYES can function as a powerful experimental workbench. *New complex algorithms.* Even with only the few elements implemented so far, we showed that algorithms approaching research-level results [4, 5, 10, 15] can be automatically derived. As more distributions, optimization methods and generalized learning algorithms are added to the system, an exponentially growing number of complex new algorithms become possible, including non-trivial variants which may challenge any single researcher's particular algorithm design expertise.

**Future agenda.** The ultimate goal is to give researchers the ability to experiment with the entire space of complex models and state-of-the-art statistical algorithms, and to allow new algorithmic ideas, as they appear, to be implicitly generalized to every model and special case known to be applicable. We have already begun work on generalizing the EM schema to continuous hidden variables, as well as adding schemas for variational methods, fast $kd$-tree and $N$-body algorithms, MCMC, and temporal models.

**Availability.** A web interface for AUTOBAYES is currently under development. More information is available at `http://ase.arc.nasa.gov/autobayes`.

## Footnotes

[1]Schema guards vary widely; for example, compare Nead-Melder simplex or simulated annealing (which require only function evaluation), conjugate gradient (which require both Jacobian and Hessian), EM and its variational extension [6] (which require a latent-variable structure model).

[2]Here, keywords have been underlined and line numbers have been added for reference in the text. The as-keyword allows annotations to variables which end up in the generated code's comments. Also, n_classes has been set to three (line 4), while n_points is left unspecified. The class variable and single data variable are vectors, which defines them as i.i.d.

[3]Popular symbolic packages such as Mathematica contain known errors allowing unsound derivations; they also lack the support for reasoning with vector and matrix quantities.

[4]Note that $U \cap descendents(U) = \emptyset$ and $U \cap parents(U) = \emptyset$.

# References

[1] C.L. Blake and C.J. Merz. UCI repository of machine learning databases, 1998.

[2] D. Blei, A.Y. Ng, and M. Jordan. Latent Dirichlet allocation. *NIPS*14*, 2002.

[3] W.L. Buntine. Operations for learning with graphical models. *JAIR*, 2:159–225, 1994.

[4] W.L. Buntine. Variational extensions to EM and multinomial PCA. *ECML 2002*, pp. 23–34, 2002.

[5] G.S. Gaffney and P. Smyth. Trajectory clustering using mixtures of regression models. In *5th KDD*, pp. 63–72, 1999.

[6] Z. Ghahramani and M.J. Beal. Propagation algorithms for variational Bayesian learning. In *NIPS*12*, pp. 507–513, 2000.

[7] P. Haddawy. Generating Bayesian Networks from Probability Logic Knowledge Bases. In *UAI 10*, pp. 262–269, 1994.

[8] F. R. Kschischang, B. Frey, and H.-A. Loeliger. Factor graphs and the sum-product algorithm. *IEEE Trans. Inform. Theory*, 47(2):498–519, 2001.

[9] S.L. Lauritzen, A.P. Dawid, B.N. Larsen, and H.-G. Leimer. Independence properties of directed Markov fields. *Networks*, 20:491–505, 1990.

[10] D.J.C. Mackay. Bayesian interpolation. *Neural Computation*, 4(3):415–447, 1991.

[11] E. Mjolsness and M. Turmon. Stochastic parameterized grammars for Bayesian model composition. In *NIPS*2000 Workshop on Software Support for Bayesian Analysis Systems*, Breckenridge, December 2000.

[12] K. Murphy. Bayes Net Toolbox for Matlab. *Interface of Computing Science and Statistics 33*, 2001.

[13] P. Smyth, D. Heckerman, and M. Jordan. Probabilistic independence networks for hidden Markov models. *Neural Computation*, 9(2):227–269, 1997.

[14] A. Thomas, D.J. Spiegelhalter, and W.R. Gilks. BUGS: A program to perform Bayesian inference using Gibbs sampling. In *Bayesian Statistics 4*, pp. 837–842, 1992.

[15] D.A. van Dyk. The nested EM algorithm. *Statistica Sinica*, 10:203-225, 2000.
